# Large margin classifiers: convex loss, low noise, and convergence rates

**Peter L. Bartlett, Michael I. Jordan and Jon D. McAuliffe**
Division of Computer Science and Department of Statistics
University of California, Berkeley
Berkeley, CA 94720
{bartlett,jordan,jon}@stat.berkeley.edu

## Abstract

Many classification algorithms, including the support vector machine, boosting and logistic regression, can be viewed as minimum contrast methods that minimize a convex surrogate of the 0-1 loss function. We characterize the statistical consequences of using such a surrogate by providing a general quantitative relationship between the risk as assessed using the 0-1 loss and the risk as assessed using any nonnegative surrogate loss function. We show that this relationship gives nontrivial bounds under the weakest possible condition on the loss function—that it satisfy a pointwise form of Fisher consistency for classification. The relationship is based on a variational transformation of the loss function that is easy to compute in many applications. We also present a refined version of this result in the case of low noise. Finally, we present applications of our results to the estimation of convergence rates in the general setting of function classes that are scaled hulls of a finite-dimensional base class.

## 1 Introduction

Convexity has played an increasingly important role in machine learning in recent years, echoing its growing prominence throughout applied mathematics (Boyd and Vandenberghe, 2003). In particular, a wide variety of two-class classification methods choose a real-valued classifier $f$ based on the minimization of a convex surrogate $\phi(yf(x))$ in the place of an intractable loss function $1(\text{sign}(f(x)) \neq y)$. Examples of this tactic include the support vector machine, AdaBoost, and logistic regression, which are based on the exponential loss, the hinge loss and the logistic loss, respectively.

What are the statistical consequences of choosing models and estimation procedures so as to exploit the computational advantages of convexity? In the setting of 0-1 loss, some basic answers have begun to emerge. In particular, it is possible to demonstrate the Bayes-risk consistency of methods based on minimizing convex surrogates for 0-1 loss, with appropriate regularization. Lugosi and Vayatis (2003) have provided such a result for any differentiable, monotone, strictly convex loss function $\phi$ that satisfies $\phi(0) = 1$. This handles many common cases although it does not handle the SVM. Steinwart (2002) has demonstrated consistency for the SVM as well, where $\mathcal{F}$ is a reproducing kernel Hilbert space and $\phi$ is continuous. Other results on Bayes-risk consistency have been presented by Jiang (2003), Zhang (2003), and Mannor et al. (2002).

To carry this agenda further, it is necessary to find general quantitative relationships between the approximation and estimation errors associated with $\phi$, and those associated with 0-1 loss. This point has been emphasized by Zhang (2003), who has presented several examples of such relationships. We simplify and extend Zhang's results, developing a general methodology for finding quantitative relationships between the risk associated with $\phi$ and the risk associated with 0-1 loss. In particular, let $R(f)$ denote the risk based on 0-1 loss and let $R^* = \inf_f R(f)$ denote the Bayes risk. Similarly, let us refer to $R_\phi(f) = \mathbf{E}\phi(Yf(X))$ as the "$\phi$-risk," and let $R_\phi^* = \inf_f R_\phi(f)$ denote the "optimal $\phi$-risk." We show that, for all measurable $f$,

$$\psi(R(f) - R^*) \leq R_\phi(f) - R_\phi^*, \tag{1}$$

for a nondecreasing function $\psi : [0,1] \rightarrow [0,\infty)$, and that no better bound is possible. Moreover, we present a general variational representation of $\psi$ in terms of $\phi$, and show how this representation allows us to infer various properties of $\psi$.

This result suggests that if $\psi$ is well-behaved then minimization of $R_\phi(f)$ may provide a reasonable surrogate for minimization of $R(f)$. Moreover, the result provides a quantitative way to transfer assessments of statistical error in terms of "excess $\phi$-risk" $R_\phi(f) - R_\phi^*$ into assessments of error in terms of "excess risk" $R(f) - R^*$.

Although our principal goal is to understand the implications of convexity in classification, we do not impose a convexity assumption on $\phi$ at the outset. Indeed, while conditions such as convexity, continuity, and differentiability of $\phi$ are easy to verify and have natural relationships to optimization procedures, it is not immediately obvious how to relate such conditions to their statistical consequences. Thus, in Section 2 we consider the weakest possible condition on $\phi$—that it is "classification-calibrated," which is essentially a pointwise form of Fisher consistency for classification. We show that minimizing $\phi$-risk leads to minimal risk precisely when $\phi$ is classification-calibrated.

Building on (1), in Section 3 we study the low noise setting, in which the posterior probability $\eta(X)$ is not too close to $1/2$. We show that in this setting we are able to obtain an improvement in the relationship between excess $\phi$-risk and excess risk.

Section 4 turns to the estimation of convergence rates for empirical $\phi$-risk minimization in the low noise setting. We find that for convex $\phi$ satisfying a certain uniform convexity condition, empirical $\phi$-risk minimization yields convergence of misclassification risk to that of the best-performing classifier in $\mathcal{F}$, and the rate of convergence can be strictly faster than the classical parametric rate of $n^{-1/2}$.

## 2 Relating excess risk to excess $\phi$-risk

There are three sources of error to be considered in a statistical analysis of classification problems: the classical estimation error due to finite sample size, the classical approximation error due to the size of the function space $\mathcal{F}$, and an additional source of approximation error due to the use of a surrogate in place of the 0-1 loss function. It is this last source of error that is our focus in this section. We give estimates for this error that are valid for any measurable function. Since the error is defined in terms of the probability distribution, we work with population expectations in this section.

Fix an input space $\mathcal{X}$ and let $(X,Y),(X_1,Y_1),\ldots,(X_n,Y_n) \in \mathcal{X} \times \{\pm 1\}$ be i.i.d., with distribution $P$. Define $\eta : \mathcal{X} \rightarrow [0,1]$ as $\eta(x) = P(Y = 1|X = x)$.

Define the $\{0,1\}$-*risk*, or just *risk*, of $f$ as $R(f) = P(\text{sign}(f(X)) \neq Y)$, where $\text{sign}(\alpha) = 1$ for $\alpha > 0$ and $-1$ otherwise. Based on the sample $D_n = ((X_1,Y_1),\ldots,(X_n,Y_n))$, we want to choose a function $f_n$ with small risk. Define the *Bayes risk* $R^* = \inf_f R(f)$, where the infimum is over all measurable $f$. Then any $f$ satisfying $\text{sign}(f(X)) = \text{sign}(\eta(X) - 1/2)$ a.s. on $\{\eta(X) \neq 1/2\}$ has $R(f) = R^*$.

Fix a function $\phi : \mathbb{R} \to [0, \infty)$. Define the $\phi$-*risk* of $f$ as $R_\phi(f) = \mathbf{E}\phi(Yf(X))$. We can view $\phi$ as specifying a contrast function that is minimized in determining a discriminant $f$. Define $C_\eta(\alpha) = \eta\phi(\alpha) + (1 - \eta)\phi(-\alpha)$, so that the *conditional $\phi$-risk* at $x \in \mathcal{X}$ is

$$\mathbf{E}(\phi(Yf(X))|X = x) = C_{\eta(x)}(f(x)) = \eta(x)\phi(f(x)) + (1 - \eta(x))\phi(-f(x)).$$

As a useful illustration for the definitions that follow, consider a singleton domain $X = \{x_0\}$. Minimizing $\phi$-risk corresponds to choosing $f(x_0)$ to minimize $C_{\eta(x_0)}(f(x_0))$.

For $\eta \in [0, 1]$, define the *optimal conditional $\phi$-risk*

$$H(\eta) = \inf_{\alpha \in \mathbb{R}} C_\eta(\alpha) = \inf_{\alpha \in \mathbb{R}} (\eta\phi(\alpha) + (1 - \eta)\phi(-\alpha)).$$

Then the *optimal $\phi$-risk* satisfies $R_\phi^* := \inf_f R_\phi(f) = \mathbf{E}H(\eta(X))$, where the infimum is over measurable functions. For $\eta \in [0, 1]$, define

$$H^-(\eta) = \inf_{\alpha : \alpha(2\eta-1)\leq 0} C_\eta(\alpha) = \inf_{\alpha : \alpha(2\eta-1)\leq 0} (\eta\phi(\alpha) + (1 - \eta)\phi(-\alpha)).$$

This is the optimal value of the conditional $\phi$-risk, under the constraint that the sign of the argument $\alpha$ disagrees with that of $2\eta - 1$.

We now turn to the basic condition we impose on $\phi$. This condition generalizes the requirement that the minimizer of $C_\eta(\alpha)$ (if it exists) has the correct sign. This is a minimal condition that can be viewed as a form of Fisher consistency for classification (Lin, 2001).

**Definition 1.** We say that $\phi$ is *classification-calibrated* if, for any $\eta \neq 1/2$,

$$H^-(\eta) > H(\eta).$$

The following functional transform of the loss function will be useful in our main result.

**Definition 2.** We define the $\psi$-*transform* of a loss function as follows. Given $\phi : \mathbb{R} \to [0, \infty)$, define the function $\psi : [0, 1] \to [0, \infty)$ by $\psi = \tilde{\psi}^{**}$, where

$$\tilde{\psi}(\theta) = H^-\left(\frac{1+\theta}{2}\right) - H\left(\frac{1+\theta}{2}\right),$$

and $g^{**} : [0, 1] \to \mathbb{R}$ is the Fenchel-Legendre biconjugate of $g : [0, 1] \to \mathbb{R}$. Equivalently, the epigraph of $g^{**}$ is the closure of the convex hull of the epigraph of $g$. (Recall that the epigraph of a function $g$ is the set $\{(x, t) : x \in [0, 1], g(x) \leq t\}$.)

It is immediate from the definitions that $\tilde{\psi}$ and $\psi$ are nonnegative and that they are also continuous on $[0, 1]$. We calculate the $\psi$-transform for exponential loss, logistic loss, quadratic loss and truncated quadratic loss, tabulating the results in Table 1. All of these loss functions can be verified to be classification-calibrated. (The other parameters listed in the table will be referred to later.)

The importance of the $\psi$-transform is shown by the following theorem.

|  | $\phi(\alpha)$ | $\psi(\theta)$ | $L_B$ | $\delta(\epsilon)$ |
|---|---|---|---|---|
| exponential | $e^{-\alpha}$ | $1 - \sqrt{1 - \theta^2}$ | $e^B$ | $e^{-B}\epsilon^2/8$ |
| logistic | $\ln(1 + e^{-2\alpha})$ | $\theta$ | $2$ | $e^{-2B}\epsilon^2/4$ |
| quadratic | $(1 - \alpha)^2$ | $\theta^2$ | $2(B+1)$ | $\epsilon^2/4$ |
| truncated quadratic | $(\max\{0, 1 - \alpha\})^2$ | $\theta^2$ | $2(B+1)$ | $\epsilon^2/4$ |

Table 1: Four convex loss functions and the corresponding $\psi$-transform. On the interval $[-B, B]$, each loss function has the indicated Lipschitz constant $L_B$ and modulus of convexity $\delta(\epsilon)$ with respect to $d_\phi$. All have a quadratic modulus of convexity.

**Theorem 3.**   *1. For any nonnegative loss function $\phi$, any measurable $f : \mathcal{X} \to \mathbb{R}$ and any probability distribution on $\mathcal{X} \times \{\pm1\}$,*

$$\psi(R(f) - R^*) \leq R_\phi(f) - R_\phi^*.$$

*2. Suppose $|\mathcal{X}| \geq 2$. For any nonnegative loss function $\phi$, any $\epsilon > 0$ and any $\theta \in [0, 1]$, there is a probability distribution on $\mathcal{X} \times \{\pm1\}$ and a function $f : \mathcal{X} \to \mathbb{R}$ such that $R(f) - R^* = \theta$ and $\psi(\theta) \leq R_\phi(f) - R_\phi^* \leq \psi(\theta) + \epsilon$.*

*3. The following conditions are equivalent.*
   *(a) $\phi$ is classification-calibrated.*
   *(b) For any sequence $(\theta_i)$ in $[0, 1]$, $\psi(\theta_i) \to 0$ if and only if $\theta_i \to 0$.*
   *(c) For every sequence of measurable functions $f_i : \mathcal{X} \to \mathbb{R}$ and every probability distribution on $\mathcal{X} \times \{\pm1\}$, $R_\phi(f_i) \to R_\phi^*$ implies $R(f_i) \to R^*$.*

**Remark**: It can be shown that classification-calibration implies $\psi$ is invertible on $[0, 1]$, in which case it is meaningful to write the upper bound on excess risk as $\psi^{-1}(R_\phi(f) - R_\phi^*)$.

**Remark**: Zhang (2003) has given a comparison theorem like Part 1, for convex $\phi$ that satisfy certain conditions. Lugosi and Vayatis (2003) and Steinwart (2002) have shown limiting results like Part 3c under other conditions on $\phi$. All of these conditions are stronger than the ones we assume here.

The following lemma summarizes various useful properties of $H$, $H^-$ and $\psi$.

**Lemma 4.** *The functions $H$, $H^-$ and $\psi$ have the following properties, for all $\eta \in [0, 1]$:*

1. *$H$ and $H^-$ are symmetric about $1/2$: $H(\eta) = H(1 - \eta)$, $H^-(\eta) = H^-(1 - \eta)$.*

2. *$H$ is concave and satisfies $H(\eta) \leq H(1/2) = H^-(1/2)$.*

3. *If $\phi$ is classification-calibrated, then $H(\eta) < H(1/2)$ for $\eta \neq 1/2$.*

4. *$H^-$ is concave on $[0, 1/2]$ and $[1/2, 1]$, and satisfies $H^-(\eta) \geq H(\eta)$.*

5. *$H$, $H^-$ and $\tilde{\psi}$ are continuous on $[0, 1]$.*

6. *$\psi$ is continuous on $[0, 1]$, $\psi$ is nonnegative and minimal at $0$, and $\psi(0) = 0$.*

7. *$\phi$ is classification-calibrated iff $\psi(\theta) > 0$ for all $\theta \in (0, 1]$.*

*Proof.* (Of Theorem 3). For Part 1, it is straightforward to show that

$$R(f) - R^* = \mathbf{E}\left(\mathbf{1}\left[\operatorname{sign}(f(X)) \neq \operatorname{sign}(\eta(X) - 1/2)\right] |2\eta(X) - 1|\right),$$

where $\mathbf{1}[\Phi]$ is $1$ if the predicate $\Phi$ is true and $0$ otherwise. From the definition, $\psi$ is convex, so we can apply Jensen's inequality, the fact that $\psi(0) = 0$ (Lemma 4, part 6) and the fact that $\psi(\theta) \leq \tilde{\psi}(\theta)$, to show that

$\psi(R(f) - R^*)$
$\leq \mathbf{E}\psi\left(\mathbf{1}\left[\operatorname{sign}(f(X)) \neq \operatorname{sign}(\eta(X) - 1/2)\right] |2\eta(X) - 1|\right)$
$= \mathbf{E}\left(\mathbf{1}\left[\operatorname{sign}(f(X)) \neq \operatorname{sign}(\eta(X) - 1/2)\right] \psi\left(|2\eta(X) - 1|\right)\right)$
$\leq \mathbf{E}\left(\mathbf{1}\left[\operatorname{sign}(f(X)) \neq \operatorname{sign}(\eta(X) - 1/2)\right] \tilde{\psi}\left(|2\eta(X) - 1|\right)\right)$
$= \mathbf{E}\left(\mathbf{1}\left[\operatorname{sign}(f(X)) \neq \operatorname{sign}(\eta(X) - 1/2)\right] \left(H^-(\eta(X)) - H(\eta(X))\right)\right)$
$= \mathbf{E}\left(\mathbf{1}\left[\operatorname{sign}(f(X)) \neq \operatorname{sign}(\eta(X) - 1/2)\right] \left(\inf_{\alpha:\alpha(2\eta(X)-1)\leq 0} C_{\eta(X)}(\alpha) - H(\eta(X))\right)\right)$
$\leq \mathbf{E}\left(C_{\eta(X)}(f(X)) - H(\eta(X))\right)$
$= R_\phi(f) - R_\phi^*,$

where the last inequality used the fact that for any $x$, and in particular when $\text{sign}(f(x)) = \text{sign}(\eta(x) - 1/2)$, we have $C_{\eta(x)}(f(x)) \geq H(\eta(x))$.

For Part 2, the first inequality is from Part 1. For the second, fix $\epsilon > 0$ and $\theta \in [0, 1]$. From the definition of $\psi$, we can choose $\gamma, \alpha_1, \alpha_2 \in [0, 1]$ for which $\theta = \gamma\alpha_1 + (1 - \gamma)\alpha_2$ and $\psi(\theta) \geq \gamma\tilde{\psi}(\alpha_1) + (1 - \gamma)\tilde{\psi}(\alpha_2) - \epsilon/2$. Choose distinct $x_1, x_2 \in \mathcal{X}$, and choose $P_X$ such that $P_X\{x_1\} = \gamma$, $P_X\{x_2\} = 1 - \gamma$, $\eta(x_1) = (1 + \alpha_1)/2$, and $\eta(x_2) = (1 + \alpha_2)/2$. From the definition of $H^-$, we can choose $f : \mathcal{X} \to \mathbb{R}$ such that $f(x_1) \leq 0$, $f(x_2) \leq 0$, $C_{\eta(x_1)}(f(x_1)) \leq H^-(\eta(x_1)) + \epsilon/2$ and $C_{\eta(x_2)}(f(x_2)) \leq H^-(\eta(x_2)) + \epsilon/2$. Then it is easy to verify that $R_\phi(f) - R_\phi^* \leq \gamma\tilde{\psi}(\alpha_1) + (1 - \gamma)\tilde{\psi}(\alpha_2) + \epsilon/2 \leq \psi(\theta) + \epsilon$. Furthermore, since $\text{sign}(f(x_i)) = -1$ but $\eta(x_i) \geq 1/2$, we have $R(f) - R^* = \mathbf{E}|2\eta(X) - 1| = \theta$.

For Part 3, first note that, for any $\phi$, $\psi$ is continuous on $[0, 1]$ and $\psi(0) = 0$ by Lemma 4, part 6, and hence $\theta_i \to 0$ implies $\psi(\theta_i) \to 0$. Thus, we can replace condition (3b) by

> (3b') For any sequence $(\theta_i)$ in $[0, 1]$, $\psi(\theta_i) \to 0$ implies $\theta_i \to 0$ .

To see that (3a) implies (3b'), let $\phi$ be classification-calibrated, and let $(\theta_i)$ be a sequence that does not converge to 0. Define $c = \limsup \theta_i > 0$, and pass to a subsequence with $\lim \theta_i = c$. Then $\lim \psi(\theta_i) = \psi(c)$ by continuity, and $\psi(c) > 0$ by classification-calibration (Lemma 4, part 7). Thus, for the original sequence $(\theta_i)$, we see $\limsup \psi(\theta_i) > 0$, so we cannot have $\psi(\theta_i) \to 0$.

Part 1 implies that (3b') implies (3c). The proof that (3c) implies (3a) is straightforward; see Bartlett et al. (2003). □

The following observation is easy to verify. It shows that if $\phi$ is convex, the classification-calibration condition is easy to verify and the $\psi$ transform is a little easier to compute.

**Lemma 5.** *Suppose $\phi$ is convex. Then we have*

   1. *$\phi$ is classification-calibrated if and only if it is differentiable at 0 and $\phi'(0) < 0$.*
   2. *If $\phi$ is classification-calibrated, then $\tilde{\psi}$ is convex, hence $\psi = \tilde{\psi}$.*

All of the classification procedures mentioned in earlier sections utilize surrogate loss functions which are either upper bounds on 0-1 loss or can be transformed into upper bounds via a positive scaling factor. It is easy to verify that this is necessary.

**Lemma 6.** *If $\phi : \mathbb{R} \to [0, \infty)$ is classification-calibrated, then there is a $\gamma > 0$ such that $\gamma\phi(\alpha) \geq \mathbf{1}\left[\alpha \leq 0\right]$ for all $\alpha \in \mathbb{R}$.*

## 3 Tighter bounds under low noise conditions

In a study of the convergence rate of empirical risk minimization, Tsybakov (2001) provided a useful condition on the behavior of the posterior probability near the optimal decision boundary $\{x : \eta(x) = 1/2\}$. Tsybakov's condition is useful in our setting as well; as we show in this section, it allows us to obtain a refinement of Theorem 3.

Recall that

$$R(f) - R^* = \mathbf{E}\left(\mathbf{1}\left[\text{sign}(f(X)) \neq \text{sign}(\eta(X) - 1/2)\right]|2\eta(X) - 1|\right)$$
$$\leq P_X\left(\text{sign}(f(X)) \neq \text{sign}(\eta(X) - 1/2)\right), \tag{2}$$

with equality provided that $\eta(X)$ is almost surely either 1 or 0. We say that $P$ *has noise exponent* $\alpha \geq 0$ if there is a $c > 0$ such that every measurable $f : \mathcal{X} \to \mathbb{R}$ has

$$P_X\left(\text{sign}(f(X)) \neq \text{sign}(\eta(X) - 1/2)\right) \leq c\left(R(f) - R^*\right)^\alpha. \tag{3}$$

Notice that we must have $\alpha \leq 1$, in view of (2). If $\alpha = 0$, this imposes no constraint on the noise: take $c = 1$ to see that every probability measure $P$ satisfies (3). On the other hand, it is easy to verify that $\alpha = 1$ if and only if $|2\eta(X) - 1| \geq 1/c$ a.s. $[P_X]$.

**Theorem 7.** *Suppose $P$ has noise exponent $0 < \alpha \leq 1$, and $\phi$ is classification-calibrated. Then there is a $c > 0$ such that for any $f : \mathcal{X} \to \mathbb{R}$,*

$$c \left(R(f) - R^*\right)^\alpha \, \psi \left(\frac{(R(f) - R^*)^{1-\alpha}}{2c}\right) \leq R_\phi(f) - R_\phi^*.$$

*Furthermore, this never gives a worse rate than the result of Theorem 3, since*

$$(R(f) - R^*)^\alpha \, \psi \left(\frac{(R(f) - R^*)^{1-\alpha}}{2c}\right) \geq \psi \left(\frac{R(f) - R^*}{2c}\right).$$

The proof follows closely that of Theorem 3(1), with the modification that we approximate the error integral separately over subsets of the input space with low and high noise.

## 4 Estimation rates

Large margin algorithms choose $\hat{f}$ from a class $\mathcal{F}$ to minimize empirical $\phi$-risk,

$$\hat{R}_\phi(f) = \hat{\mathbf{E}}\phi(Yf(X)) = \frac{1}{n} \sum_{i=1}^{n} \phi(Y_i f(X_i)).$$

We have seen how the excess risk depends on the excess $\phi$-risk. In this section, we examine the convergence of $\hat{f}$'s excess $\phi$-risk, $R_\phi(\hat{f}) - R_\phi^*$. We can split this excess risk into an estimation error term and an approximation error term:

$$R_\phi(\hat{f}) - R_\phi^* = (R_\phi(\hat{f}) - \inf_{f \in \mathcal{F}} R_\phi(f)) + (\inf_{f \in \mathcal{F}} R_\phi(f) - R_\phi^*).$$

We focus on the first term, the estimation error term. For simplicity, we assume throughout that some $f^* \in \mathcal{F}$ achieves the infimum, $R_\phi(f^*) = \inf_{f \in \mathcal{F}} R_\phi(f)$.

The simplest way to bound $R_\phi(\hat{f}) - R_\phi(f^*)$ is to show that $\hat{R}_\phi(f)$ and $R_\phi(f)$ are close, uniformly over $\mathcal{F}$. This approach can give the wrong rate. For example, for a nontrivial class $\mathcal{F}$, the resulting estimation error bound can decrease no faster than $1/\sqrt{n}$. However, if $\mathcal{F}$ is a small class (for instance, a VC-class) and $R_\phi(f^*) = 0$, then $R_\phi(\hat{f})$ should decrease as $\log n/n$. Lee et al. (1996) showed that fast rates are also possible for the quadratic loss $\phi(\alpha) = (1 - \alpha)^2$ if $\mathcal{F}$ is convex, even if $R_\phi(f^*) > 0$. In particular, because the quadratic loss function is strictly convex, it is possible to bound the variance of the excess loss (difference between the loss of a function $f$ and that of the optimal $f^*$) in terms of its expectation. Since the variance decreases as we approach the optimal $f^*$, the risk of the empirical minimizer converges more quickly to the optimal risk than the simple uniform convergence results would suggest. Mendelson (2002) improved this result, and extended it from prediction in $L_2(P_X)$ to prediction in $L_p(P_X)$ for other values of $p$. The proof used the idea of the modulus of convexity of a norm. This idea can be used to give a simpler proof of a more general bound when the loss function satisfies a strict convexity condition, and we obtain risk bounds. The modulus of convexity of an arbitrary strictly convex function (rather than a norm) is a key notion in formulating our results.

**Definition 8 (Modulus of convexity).** Given a pseudometric $d$ defined on a vector space $S$, and a convex function $f : S \to \mathbb{R}$, the *modulus of convexity* of $f$ with respect to $d$ is the function $\delta : [0, \infty) \to [0, \infty]$ satisfying

$$\delta(\epsilon) = \inf \left\{ \frac{f(x_1) + f(x_2)}{2} - f\left(\frac{x_1 + x_2}{2}\right) : x_1, x_2 \in S, \, d(x_1, x_2) \geq \epsilon \right\}.$$

If $\delta(\epsilon) > 0$ for all $\epsilon > 0$, we say that $f$ is *strictly convex* with respect to $d$.

We consider loss functions $\phi$ that also satisfy a Lipschitz condition with respect to a pseudometric $d$ on $\mathbb{R}$: we say that $\phi : \mathbb{R} \to \mathbb{R}$ is Lipschitz with respect to $d$, with constant $L$, if for all $a, b \in \mathbb{R}$, $|\phi(a) - \phi(b)| \leq L \cdot d(a, b)$. (Note that if $d$ is a metric and $\phi$ is convex, then $\phi$ necessarily satisfies a Lipschitz condition on any compact subset of $\mathbb{R}$.)

We consider four loss functions that satisfy these conditions: the exponential loss function used in AdaBoost, the deviance function for logistic regression, the quadratic loss function, and the truncated quadratic loss function; see Table 1. We use the pseudometric

$$d_\phi(a, b) = \inf \left\{ |a - \alpha| + |\beta - b| : \ \phi \text{ constant on } (\min\{\alpha, \beta\}, \max\{\alpha, \beta\}) \right\}.$$

For all except the truncated quadratic loss function, this corresponds to the standard metric on $\mathbb{R}$, $d_\phi(a, b) = |a - b|$. In all cases, $d_\phi(a, b) \leq |a - b|$, but for the truncated quadratic, $d_\phi$ ignores differences to the right of $1$. It is easy to calculate the Lipschitz constant and modulus of convexity for each of these loss functions. These parameters are given in Table 1.

In the following result, we consider the function class used by algorithms such as AdaBoost: the class of linear combinations of classifiers from a fixed base class. We assume that this base class has finite Vapnik-Chervonenkis dimension, and we constrain the size of the class by restricting the $\ell_1$ norm of the linear parameters. If $\mathcal{G}$ is the VC-class, we write $\mathcal{F} = B\operatorname{absconv}(\mathcal{G})$, for some constant $B$, where

$$B\operatorname{absconv}(\mathcal{G}) = \left\{ \sum_{i=1}^{m} \alpha_i g_i : m \in \mathbb{N}, \ \alpha_i \in \mathbb{R}, \ g_i \in \mathcal{G}, \ \|\alpha\|_1 = B \right\}.$$

**Theorem 9.** *Let $\phi : \mathbb{R} \to \mathbb{R}$ be a convex loss function. Suppose that, on the interval $[-B, B]$, $\phi$ is Lipschitz with constant $L_B$ and has modulus of convexity $\delta(\epsilon) = a_B \epsilon^2$ (both with respect to the pseudometric $d$).*

*For any probability distribution $P$ on $\mathcal{X} \times \mathcal{Y}$ that has noise exponent $\alpha = 1$, there is a constant $c'$ for which the following is true. For i.i.d. data $(X_1, Y_1), \ldots, (X_n, Y_n)$, let $\hat{f} \in \mathcal{F}$ be the minimizer of the empirical $\phi$-risk, $R_\phi(f) = \hat{\mathbf{E}}\phi(Yf(X))$. Suppose that $\mathcal{F} = B\operatorname{absconv}(\mathcal{G})$, where $\mathcal{G} \subseteq \{\pm 1\}^{\mathcal{X}}$ has $d_{VC}(\mathcal{G}) = d$, and*

$$\epsilon^* \geq BL_B \max \left\{ \left( \frac{L_B a_B}{B} \right)^{1/(d+1)}, 1 \right\} n^{-(d+2)/(2d+2)}$$

*Then with probability at least $1 - e^{-x}$,*

$$R(\hat{f}) \leq R^* + c' \left( \epsilon^* + \frac{L_B(L_B/a_B + B)x}{n} + \inf_{f \in \mathcal{F}} R_\phi(f) - R_\phi^* \right).$$

Notice that the rate obtained here is strictly faster than the classical $n^{-1/2}$ parametric rate, even though the class is infinite dimensional and the optimal element of $\mathcal{F}$ can have risk larger than the Bayes risk. The key idea in the proof is similar to ideas from Lee et al. (1996), Mendelson (2002), but simpler. Let $f^*$ be the minimizer of $\phi$-risk in a function class $\mathcal{F}$. If the class $\mathcal{F}$ is convex and the loss function $\phi$ is strictly convex and Lipschitz, then the variance of the excess loss, $g_f(x, y) = \phi(yf(x)) - \phi(yf^*(x))$, decreases with its expectation. Thus, as a function $f \in \mathcal{F}$ approaches the optimum, $f^*$, the two losses $\phi(Y\hat{f}(X))$ and $\phi(Yf^*(X))$ become strongly correlated. This leads to the faster rates. More formally, suppose that $\phi$ is $L$-Lipschitz and has modulus of convexity $\delta(\epsilon) \geq c\epsilon^r$ with $r \leq 2$. Then it is straightforward to show that $\mathbf{E}g_f^2 \leq L^2 \left( \mathbf{E}g_f/(2c) \right)^{2/r}$. For the details, see Bartlett et al. (2003).

## 5  Conclusions

We have studied the relationship between properties of a nonnegative margin-based loss function $\phi$ and the statistical performance of the classifier which, based on an i.i.d. training

set, minimizes empirical $\phi$-risk over a class of functions. We first derived a universal upper bound on the population misclassification risk of any thresholded measurable classifier in terms of its corresponding population $\phi$-risk. The bound is governed by the $\psi$-transform, a convexified variational transform of $\phi$. It is the tightest possible upper bound uniform over all probability distributions and measurable functions in this setting.

Using this upper bound, we characterized the class of loss functions which guarantee that every $\phi$-risk consistent classifier sequence is also Bayes-risk consistent, under any population distribution. Here $\phi$-risk consistency denotes sequential convergence of population $\phi$-risks to the smallest possible $\phi$-risk of any measurable classifier. The characteristic property of such a $\phi$, which we term classification-calibration, is a kind of pointwise Fisher consistency for the conditional $\phi$-risk at each $x \in \mathcal{X}$. The necessity of classification-calibration is apparent; the sufficiency underscores its fundamental importance in elaborating the statistical behavior of large-margin classifiers.

Under the low noise assumption of Tsybakov (2001), we sharpened our original upper bound and studied the Bayes-risk consistency of $\hat{f}$, the minimizer of empirical $\phi$-risk over a convex, bounded class of functions $\mathcal{F}$ which is not too complex. We found that, for convex $\phi$ satisfying a certain uniform strict convexity condition, empirical $\phi$-risk minimization yields convergence of misclassification risk to that of the best-performing classifier in $\mathcal{F}$, as the sample size grows. Furthermore, the rate of convergence can be strictly faster than the classical $n^{-1/2}$, depending on the strictness of convexity of $\phi$ and the complexity of $\mathcal{F}$.

### Acknowledgments

We would like to thank Gilles Blanchard, Olivier Bousquet, Pascal Massart, Ron Meir, Shahar Mendelson, Martin Wainwright and Bin Yu for helpful discussions.

# References

Bartlett, P. L., Jordan, M. I., and McAuliffe, J. M. (2003). Convexity, classification and risk bounds. Technical Report 638, Dept. of Statistics, UC Berkeley. [www.stat.berkeley.edu/tech-reports].

Boyd, S. and Vandenberghe, L. (2003). *Convex Optimization*. [www.stanford.edu/∼boyd].

Jiang, W. (2003). Process consistency for Adaboost. *Annals of Statistics,* in press.

Lee, W. S., Bartlett, P. L., and Williamson, R. C. (1996). Efficient agnostic learning of neural networks with bounded fan-in. *IEEE Transactions on Information Theory*, 42(6):2118–2132.

Lin, Y. (2001). A note on margin-based loss functions in classification. Technical Report 1044r, Department of Statistics, University of Wisconsin.

Lugosi, G. and Vayatis, N. (2003). On the Bayes risk consistency of regularized boosting methods. *Annals of Statistics,* in press.

Mannor, S., Meir, R., and Zhang, T. (2002). The consistency of greedy algorithms for classification. In *Proceedings of the Annual Conference on Computational Learning Theory*, pages 319–333.

Mendelson, S. (2002). Improving the sample complexity using global data. *IEEE Transactions on Information Theory*, 48(7):1977–1991.

Steinwart, I. (2002). Consistency of support vector machines and other regularized classifiers. Technical Report 02-03, University of Jena, Department of Mathematics and Computer Science.

Tsybakov, A. (2001). Optimal aggregation of classifiers in statistical learning. Technical Report PMA-682, Université Paris VI.

Zhang, T. (2003). Statistical behavior and consistency of classification methods based on convex risk minimization. *Annals of Statistics,* in press.
